# Statistical Consistency of Top-$k$ Ranking

**Fen Xia**
Institute of Automation
Chinese Academy of Sciences
fen.xia@ia.ac.cn

**Tie-Yan Liu**
Microsoft Research Asia
tyliu@microsoft.com

**Hang Li**
Microsoft Research Asia
hanglig@microsoft.com

## Abstract

This paper is concerned with the consistency analysis on listwise ranking methods. Among various ranking methods, the listwise methods have competitive performances on benchmark datasets and are regarded as one of the state-of-the-art approaches. Most listwise ranking methods manage to optimize ranking on the whole list (permutation) of objects, however, in practical applications such as information retrieval, correct ranking at the top $k$ positions is much more important. This paper aims to analyze whether existing listwise ranking methods are statistically consistent in the top-$k$ setting. For this purpose, we define a top-$k$ ranking framework, where the true loss (and thus the risks) are defined on the basis of top-$k$ subgroup of permutations. This framework can include the permutation-level ranking framework proposed in previous work as a special case. Based on the new framework, we derive sufficient conditions for a listwise ranking method to be consistent with the top-$k$ true loss, and show an effective way of modifying the surrogate loss functions in existing methods to satisfy these conditions. Experimental results show that after the modifications, the methods can work significantly better than their original versions.

## 1 Introduction

Ranking is the central problem in many applications including information retrieval (IR). In recent years, machine learning technologies have been successfully applied to ranking, and many learning to rank methods have been proposed, including the pointwise [12] [9] [6], pairwise [8] [7] [2], and listwise methods [13] [3] [16]. Empirical results on benchmark datasets have demonstrated that the listwise ranking methods have very competitive ranking performances [10].

To explain the high ranking performances of the listwise ranking methods, a theoretical framework was proposed in [16]. In the framework, existing listwise ranking methods are interpreted as making use of different surrogate loss functions of the permutation-level 0-1 loss. Theoretical analysis shows that these surrogate loss functions are all statistically consistent in the sense that minimization of the conditional expectation of them will lead to obtaining the Bayes ranker, i.e., the optimal ranked list of the objects.

Here we point out that there is a gap between the analysis in [16] and many real ranking problems, where the correct ranking of the entire permutation is not needed. For example, in IR, users usually care much more about the top ranking results and thus only correct ranking at the top positions is important. In this new situation, it is no longer clear whether existing listwise ranking methods are still statistically consistent. The motivation of this work is to perform formal study on the issue.

For this purpose, we propose a new ranking framework, in which the "true loss" is defined on the top-$k$ subgroup of permutations instead of on the entire permutation. The new true loss only measures errors occurring at the top $k$ positions of a ranked list, therefore we refer to it as the top-$k$ true loss (Note that when $k$ equals the length of the ranked list, the top-$k$ true loss will become exactly

the permutation-level 0-1 loss). We prove a new theorem which gives sufficient conditions for a surrogate loss function to be consistent with the top-$k$ true loss. We also investigate the change of the conditions with respect to different $k$'s. Our analysis shows that, as $k$ decreases, to guarantee the consistency of a surrogate loss function, the requirement on the probability space becomes weaker while the requirement on the surrogate loss function itself becomes stronger. As a result, a surrogate loss function that is consistent with the permutation-level 0-1 loss might not be consistent with the top-$k$ true loss any more. Therefore, the surrogate loss functions in existing listwise ranking methods, which have been proved to be consistent with the permutation-level 0-1 loss, are not theoretically guaranteed to have good performances in the top-$k$ setting. Modifications to these surrogate loss functions are needed to further make them consistent with the top-$k$ true loss. We show how to make such modifications, and empirically verify that such modifications can lead to significant performance improvement. This validates the correctness of our theoretical analysis.

## 2 Permutation-level ranking framework

We review the permutation-level ranking framework proposed in [16].

Let $X$ be the input space whose elements are groups of objects to be ranked, $Y$ be the output space whose elements are permutations of objects, and $P_{XY}$ be an unknown but fixed joint probability distribution of $X$ and $Y$. Let $h \in H : X \to Y$ be a ranking function. Let $\mathbf{x} \in X$ and $y \in Y$, and let $y(i)$ be the index of the object that is ranked at position $i$ in $y$. The task of learning to rank is to learn a function that can minimize the expected risk $R(h)$, defined as,

$$R(h) = \int_{X \times Y} l(h(\mathbf{x}), y) dP(\mathbf{x}, y), \tag{1}$$

where $l(h(\mathbf{x}), y)$ is the true loss such that

$$l(h(\mathbf{x}), y) = \begin{cases} 1, & \text{if } h(\mathbf{x}) \neq y \\ 0, & \text{if } h(\mathbf{x}) = y. \end{cases} \tag{2}$$

The above true loss indicates that if the permutation of the predicted result is exactly the same as the permutation in the ground truth, then the loss is zero; otherwise the loss is one. For ease of reference, we call it permutation-level 0-1 loss. The optimal ranking function which can minimize the expected true risk $R(h^*) = \inf R(h)$ is referred to as the permutation-level Bayes ranker.

$$h^*(\mathbf{x}) = \arg \max_{y \in Y} P(y|\mathbf{x}). \tag{3}$$

In practice, for efficiency consideration, the ranking function is usually defined as $h(\mathbf{x}) = \text{sort}(g(x_1), \ldots, g(x_n))$, where $g(\cdot)$ denotes the scoring function, and $\text{sort}(\cdot)$ denotes the sorting function. Since the risk is non-continuous and non-differentiable with respect to the scoring function $g$, a continuous and differentiable surrogate loss function $\phi(\mathbf{g}(\mathbf{x}), y)$ is usually used as an approximation of the true loss. In this way, the expected risk becomes

$$R^\phi(\mathbf{g}) = \int_{X \times Y} \phi(\mathbf{g}(\mathbf{x}), y) dP(\mathbf{x}, y), \tag{4}$$

where $\mathbf{g}(\mathbf{x}) = (g(x_1), \ldots, g(x_n))$ is a vector-valued function induced by $g$.

It has been shown in [16] that many existing listwise ranking methods fall into the above framework, with different surrogate loss functions used. Furthermore, their surrogate loss functions are statistically consistent under certain conditions with respect to the permutation-level 0-1 loss. However, as shown in the next section, the permutation-level 0-1 loss is not suitable to describe the ranking problem in many real applications.

## 3 Top-$k$ ranking framework

We next describe the real ranking problem, and then propose the top-$k$ ranking framework.

## 3.1 Top-$k$ ranking problem

In real ranking applications like IR, people pay more attention to the top-ranked objects. Therefore the correct ranking on the top positions is critically important. For example, modern web search engines only return top $1,000$ results and 10 results in each page. According to a user study[1], 62% of search engine users only click on the results within the first page, and 90% of users click on the results within the first three pages. It means that two ranked lists of documents will likely provide the same experience to the users (and thus suffer the same loss), if they have the same ranking results for the top positions. This, however, cannot be reflected in the permutation-level 0-1 loss in Eq.(2). This characteristic of ranking problems has also been explored in earlier studies in different settings [4, 5, 14]. We refer to it as the top-$k$ ranking problem.

## 3.2 Top-$k$ true loss

To better describe the top-$k$ ranking problem, we propose defining the true loss based on the top $k$ positions in a ranked list, referred to as the top-$k$ true loss.

$$l_k(h(\mathbf{x}), y) = \begin{cases} 0, & \text{if } \hat{y}(i) = y(i) \quad \forall i \in \{1, \ldots, k\}, \text{where } \hat{y} = h(\mathbf{x}), \\ 1, & \text{otherwise} . \end{cases} \tag{5}$$

The actual value of $k$ is determined by application. When $k$ equals the length of the entire ranked list, the top-$k$ true loss will become exactly the permutation-level 0-1 loss. In this regard, the top-$k$ true loss is more general than the permutation-level 0-1 loss.

With Eq.(5), the expected risk becomes

$$R_k(h) = \int_{X \times Y} l_k(h(\mathbf{x}), y) dP(\mathbf{x}, y). \tag{6}$$

It can be proved that the optimal ranking function with respect to the top-$k$ true loss (i.e., the top-$k$ Bayes ranker) is any permutation in the top-$k$ subgroup having the highest probability[2], i.e.,

$$h_k^*(\mathbf{x}) \in \arg\max_{G_k(j_1, j_2, \ldots, j_k) \in G_k} P(G_k(j_1, j_2, \ldots, j_k) | \mathbf{x}), \tag{7}$$

where $G_k(j_1, j_2, \ldots, j_k) = \{y \in Y | y(t) = j_t, \forall t = 1, 2, \ldots k\}$ denotes a top-$k$ subgroup in which all the permutations have the same top-$k$ true loss; $G_k$ denotes the collection of all top-$k$ subgroups.

With the above setting, we will analyze the consistency of the surrogate loss functions in existing ranking methods with the top-$k$ true loss in the next section.

# 4 Theoretical analysis

In this section, we first give the sufficient conditions of consistency for the top-$k$ ranking problem. Next, we show how these conditions change with respect to $k$. Last, we discuss whether the surrogate loss functions in existing methods are consistent, and how to make them consistent if not.

## 4.1 Statistical consistency

We investigate what kinds of surrogate loss functions $\phi(\mathbf{g}(\mathbf{x}), y)$ are statistically consistent with the top-$k$ true loss. For this purpose, we study whether the ranking function that minimizes the conditional expectation of the surrogate loss function defined as follows coincides with the top-$k$ Bayes ranker as defined in Eq.(7).

$$Q(P(y|\mathbf{x}), \mathbf{g}(\mathbf{x})) = \sum_{y \in Y} P(y|\mathbf{x})\phi(\mathbf{g}(\mathbf{x}), y). \tag{8}$$

According to [1], the above condition is the weakest condition to guarantee that optimizing a surrogate loss function will lead to obtaining a model achieving the Bayes risk (in our case, the top-$k$ Bayes ranker), when the training sample size approaches infinity.

We denote $Q(P(y|\mathbf{x}), \mathbf{g}(\mathbf{x}))$ as $Q(\mathbf{p}, \mathbf{g})$, $\mathbf{g}(\mathbf{x})$ as $\mathbf{g}$ and $P(y|\mathbf{x})$ as $p_y$. Hence, $Q(\mathbf{p}, \mathbf{g})$ is the loss of $\mathbf{g}$ at $\mathbf{x}$ with respect to the conditional probability distribution $p_y$. The key idea is to decompose the sorting of $\mathbf{g}$ into pairwise relationship between scores of objects. To this end, we denote $Y_{i,j}$ as a permutation set in which each permutation ranks object $i$ before object $j$, i.e., $Y_{i,j} \triangleq \{y \in Y : y^{-1}(i) < y^{-1}(j)\}$ (here $y^{-1}(j)$ denotes the position of object $j$ in permutation $y$), and introduce the following definitions.

**Definition 1.** $\Lambda_{G_k}$ is the a top-$k$ subgroup probability space, such that $\Lambda_{G_k} \triangleq \{\mathbf{p} \in R^{|G_k|} : \sum_{G_k(j_1, j_2, \ldots, j_k) \in G_k} p_{G_k(j_1, j_2, \ldots, j_k)} = 1, p_{G_k(j_1, j_2, \ldots, j_k)} \geq 0\}$.

**Definition 2.** A top-$k$ subgroup probability space $\Lambda_{G_k}$ is order preserving with respect to objects $i$ and $j$, if $\forall y \in Y_{i,j}$ and $G_k(y(1), y(2), \ldots, y(k)) \neq G_k(\sigma_{i,j}^{-1} y(1), \sigma_{i,j}^{-1} y(2), \ldots, \sigma_{i,j}^{-1} y(k))$, we have $p_{G_k(y(1), y(2), \ldots, y(k))} > p_{G_k(\sigma_{i,j}^{-1} y(1), \sigma_{i,j}^{-1} y(2), \ldots, \sigma_{i,j}^{-1} y(k))}$. Here $\sigma_{i,j}^{-1} y$ denotes the permutation in which the positions of objects $i$ and $j$ are exchanged while those of the other objects remain the same as in $y$.

**Definition 3.** A surrogate loss function $\phi$ is top-$k$ subgroup order sensitive on a set $\Omega \subset R^n$, if $\phi$ is a non-negative differentiable function and the following three conditions hold for $\forall$ objects $i$ and $j$: (1) $\phi(\mathbf{g}, y) = \phi(\sigma_{i,j}^{-1} \mathbf{g}, \sigma_{i,j}^{-1} y)$; (2) Assume $g_i < g_j$, $\forall y \in Y_{i,j}$. If $G_k(y(1), y(2), \ldots, y(k)) \neq G_k(\sigma_{i,j}^{-1} y(1), \sigma_{i,j}^{-1} y(2), \ldots, \sigma_{i,j}^{-1} y(k))$, then $\phi(\mathbf{g}, y) \geq \phi(\mathbf{g}, \sigma_{i,j}^{-1} y)$ and for at least one $y$, the strict inequality holds; otherwise, $\phi(\mathbf{g}, y) = \phi(\mathbf{g}, \sigma_{i,j}^{-1} y)$. (3) Assume $g_i = g_j$. $\exists y \in Y_{i,j}$ with $G_k(y(1), y(2), \ldots, y(k)) \neq G_k(\sigma_{i,j}^{-1} y(1), \sigma_{i,j}^{-1} y(2), \ldots, \sigma_{i,j}^{-1} y(k))$ satisfying $\frac{\partial \phi(\mathbf{g}, \sigma_{i,j}^{-1} y)}{\partial g_i} > \frac{\partial \phi(\mathbf{g}, y)}{\partial g_i}$.

The *order preserving* property of a top-$k$ subgroup probability space (see Definition 2) indicates that if the top-$k$ subgroup probability on a permutation $y \in Y_{i,j}$ is larger than that on permutation $\sigma_{i,j}^{-1} y$, then the relation holds for any other permutation $y'$ in $Y_{i,j}$ and and the corresponding $\sigma_{i,j}^{-1} y'$ provided that the top-$k$ subgroup of the former is different from that of the latter. The *order sensitive* property of a surrogate loss function (see Definition 3) indicates that (i) $\phi(\mathbf{g}, y)$ exhibits a symmetry in the sense that simultaneously exchanging the positions of objects $i$ and $j$ in the ground truth and their scores in the predicted score list will not make the surrogate loss change. (ii) When a permutation is transformed to another permutation by exchanging the positions of two objects of it, if the two permutations do not belong to the same top-$k$ subgroup, the loss on the permutation that ranks the two objects in the decreasing order of their scores will not be greater than the loss on its counterpart. (iii) There exists a permutation, for which the speed of change in loss with respect to the score of an object will become faster if exchanging its position with another object with the same score but ranked lower. A top-$k$ subgroup order sensitive surrogate loss function has several nice properties as shown below.

**Proposition 4.** Let $\phi(\mathbf{g}, y)$ be a top-$k$ subgroup order sensitive loss function. $\forall y, \forall \pi \in G_k(y(1), y(2), \ldots, y(k))$, we have $\phi(\mathbf{g}, \pi) = \phi(\mathbf{g}, y)$.

**Proposition 5.** Let $\phi(\mathbf{g}, y)$ be a top-$k$ subgroup order sensitive surrogate loss function. $\forall$ objects $i$ and $j$ with $g_i = g_j$, $\forall y \in Y_{i,j}$, if $G_k(y(1), y(2), \ldots, y(k)) \neq G_k(\sigma_{i,j}^{-1} y(1), \sigma_{i,j}^{-1} y(2), \ldots, \sigma_{i,j}^{-1} y(k))$, then $\frac{\partial \phi(\mathbf{g}, \sigma_{i,j}^{-1} y)}{\partial g_i} \geq \frac{\partial \phi(\mathbf{g}, y)}{\partial g_i}$. Otherwise, $\frac{\partial \phi(\mathbf{g}, \sigma_{i,j}^{-1} y)}{\partial g_i} = \frac{\partial \phi(\mathbf{g}, y)}{\partial g_i}$.

Proposition 4 shows that all permutations in the same top-$k$ subgroup share the same loss $\phi(\mathbf{g}, y)$ and thus share the same partial difference with respect to the score of a given object. Proposition 5 indicates that the partial difference of $\phi(\mathbf{g}, y)$ also has a similar property to $\phi(\mathbf{g}, y)$ (see the second condition in Definition 3). Due to space restriction, we omit the proofs (see [15] for more details).

Based on the above definitions and propositions, we give the main theorem (Theorem 6), which states the sufficient conditions for a surrogate loss function to be consistent with the top-$k$ true loss.

**Theorem 6.** Let $\phi$ be a top-$k$ subgroup order sensitive loss function on $\Omega \subset R^n$. For $\forall n$ objects, if its top-$k$ subgroup probability space is order preserving with respect to $n - 1$ object pairs $\{(j_i, j_{i+1})\}_{i=1}^k$ and $\{(j_{k+s_i}, j_{k+i} : 0 \leq s_i < i)\}_{i=2}^{n-k}$, then the loss $\phi(\mathbf{g}, y)$ is consistent with the top-$k$ true loss as defined in Eq.(5).

The proof of the main theorem is mostly based on Theorem 7, which specifies the score relation between two objects for the minimizer of $Q(\mathbf{p}, \mathbf{g})$. Due to space restriction, we only give Theorem 7 and its detailed proof. For the detailed proof of Theorem 6, please refer to [15].

**Theorem 7.** *Let $\phi(\mathbf{g}, y)$ be a top-$k$ subgroup order sensitive loss function. $\forall i$ and $j$, if the top-$k$ subgroup probability space is order preserving with respect to them, and $\mathbf{g}$ is a vector which minimizes $Q(\mathbf{p}, \mathbf{g})$ in Eq.(8), then $g_i > g_j$.*

*Proof.* Without loss of generality, we assume $i = 1$, $j = 2$, $g_1' = g_2$, $g_2' = g_1$, and $g_k' = g_k (k > 2)$.

First, we prove $g_1 \geq g_2$ by contradiction. Assume $g_1 < g_2$, we have

$$Q(\mathbf{p}, \mathbf{g}') - Q(\mathbf{p}, \mathbf{g}) = \sum_{y \in Y} (p_{\sigma_{1,2}^{-1}y} - p_y)\phi(\mathbf{g}, y) = \sum_{y \in Y_{1,2}} (p_{\sigma_{1,2}^{-1}y} - p_y)(\phi(\mathbf{g}, y) - \phi(\mathbf{g}, \sigma_{1,2}^{-1}y)).$$

The first equation is based on the fact $\mathbf{g}' = \sigma_{1,2}^{-1}\mathbf{g}$, and the second equation is based on the fact $\sigma_{1,2}^{-1}\sigma_{1,2}^{-1}y = y$. After some algebra, by using Proposition 4, we have,

$$Q(\mathbf{p}, \mathbf{g}') - Q(\mathbf{p}, \mathbf{g}) = \sum_{G_k(y) \in \{G_k : G_k(y) \neq G_k(\sigma_{1,2}^{-1}y)\}: y \in Y_{1,2}} (p_{G_k(\sigma_{1,2}^{-1}y)} - p_{G_k(y)})(\phi(\mathbf{g}, y) - \phi(\mathbf{g}, \sigma_{1,2}^{-1}y)),$$

where $G_k(y)$ denotes the subgroup that $y$ belongs to.

Since $g_1 < g_2$, we have $\phi(\mathbf{g}, y) \geq \phi(\mathbf{g}, \sigma_{1,2}^{-1}y)$. Meanwhile, $p_{G_k(\sigma_{1,2}^{-1}y)} < p_{G_k(y)}$ due to the order preserving of the top-$k$ subgroup probability space. Thus each component in the sum is non-positive and at least one of them is negative, which means $Q(\mathbf{p}, \mathbf{g}') < Q(\mathbf{p}, \mathbf{g})$. This is a contradiction to the optimality of $\mathbf{g}$. Therefore, we must have $g_1 \geq g_2$.

Second, we prove $g_1 \neq g_2$, again by contradiction. Assume $g_1 = g_2$. By setting the derivative of $Q(\mathbf{p}, \mathbf{g})$ with respect to $g_1$ and $g_2$ to zero and compare them[3], we have,

$$\sum_{y \in Y_{1,2}} (p_y - p_{\sigma_{1,2}^{-1}y})\left(\frac{\partial\phi(\mathbf{g}, y)}{\partial g_1} - \frac{\partial\phi(\mathbf{g}, \sigma_{1,2}^{-1}y)}{\partial g_1}\right) = 0.$$

After some algebra, we obtain,

$$\sum_{G_k(y) \in \{G_k : G_k(y) \neq G_k(\sigma_{1,2}^{-1}y)\}: y \in Y_{1,2}} (p_{G_k(y)} - p_{G_k(\sigma_{1,2}^{-1}y)})\left(\frac{\partial\phi(\mathbf{g}, y)}{\partial g_1} - \frac{\partial\phi(\mathbf{g}, \sigma_{1,2}^{-1}y)}{\partial g_1}\right) = 0.$$

According to Proposition 5, we have $\frac{\partial\phi(\mathbf{g}, y)}{\partial g_1} \leq \frac{\partial\phi(\mathbf{g}, \sigma_{1,2}^{-1}y)}{\partial g_1}$. Meanwhile, $p_{G_k(\sigma_{1,2}^{-1}y)} < p_{G_k(y)}$ due to the order preserving of the top-$k$ subgroup probability space. Thus, the above equation cannot hold since at least one of components in the sum is negative according to Definition 3. $\square$

## 4.2 Consistency with respect to $k$

We discuss the change of the consistency conditions with respect to various $k$ values.

First, we have the following proposition for the top-$k$ subgroup probability space.

**Proposition 8.** *If the top-$k$ subgroup probability space is order preserving with respect to object $i$ and $j$, the top-$(k-1)$ subgroup probability space is also order preserving with respect to $i$ and $j$.*

The proposition can be proved by decomposing a top-$(k-1)$ subgroup into the sum of top-$k$ subgroups. One can find the detailed proof in [15]. Here we give an example to illustrate the basic idea. Suppose there are three objects $\{1, 2, 3\}$ to be ranked. If the top-2 subgroup probability space is order preserving with respect to objects 1 and 2, then we have $p_{G_2(1,2)} > p_{G_2(2,1)}$, $p_{G_2(1,3)} > p_{G_2(2,3)}$ and $p_{G_2(3,1)} > p_{G_2(3,2)}$. On the other hand, for top-1, we have $p_{G_1(1)} > p_{G_1(2)}$. Note that $p_{G_1(1)} = p_{G_2(1,2)} + p_{G_2(1,3)}$ and $p_{G_1(2)} = p_{G_2(2,1)} + p_{G_2(2,3)}$. Thus, it is easy to verify that Proposition 8 holds for this case while the opposite does not.

Second, we obtain the following proposition for the surrogate loss function $\phi$.

**Proposition 9.** *If the surrogate loss function $\phi$ is top-$k$ subgroup order sensitive on a set $\Omega \subset R^n$, then it is also top-$(k+1)$ subgroup order sensitive on the same set.*

Again, one can refer to [15] for the detailed proof of the proposition, and here we only provide an example. Let us consider the same setting in the previous example. Assume that $g_1 < g_2$. If $\phi$ is top-1 subgroup order sensitive, then we have $\phi(\mathbf{g}, (1,2,3)) \geq \phi(\mathbf{g}, (2,1,3))$, $\phi(\mathbf{g}, (1,3,2)) \geq \phi(\mathbf{g}, (2,3,1))$, and $\phi(\mathbf{g}, (3,1,2)) = \phi(\mathbf{g}, (3,2,1))$. From Proposition 4, we know that the two inequalities are strict. On the other hand, if $\phi$ is top-2 subgroup order sensitive, the following inequalities hold with at least one of them being strict: $\phi(\mathbf{g}, (1,2,3)) \geq \phi(\mathbf{g}, (2,1,3))$, $\phi(\mathbf{g}, (1,3,2)) \geq \phi(\mathbf{g}, (2,3,1))$, and $\phi(\mathbf{g}, (3,1,2)) \geq \phi(\mathbf{g}, (3,2,1))$. Therefore top-1 subgroup order sensitive is a special case of top-2 subgroup order sensitive.

According to the above propositions, we can come to the following conclusions.

- For the consistency with the top-$k$ true loss, when $k$ becomes smaller, the requirement on the probability space becomes weaker but the requirement on the surrogate loss function becomes stronger. Since we never know the real property of the (unknown) probability space, it is more likely the requirement on the probability space for the consistency with the top-$k$ true loss can be satisfied than that for the top-$l$ ($l > k$) true loss. Specifically, it is risky to assume the requirement for the permutation-level 0-1 loss to hold.

- If we fix the true loss to be top-$k$ and the probability space to be top-$k$ subgroup order preserving, the surrogate loss function should be at most top-$l$ ($l \leq k$) subgroup order sensitive in order to meet the consistency conditions. It is not guaranteed that a top-$l$ ($l > k$) subgroup order sensitive surrogate loss function can be consistent with the top-$k$ true loss. For example, a top-1 subgroup order sensitive surrogate loss function may be consistent with any top-$k$ true loss, but a permutation-level order sensitive surrogate loss function may not be consistent with any top-$k$ true loss, if $k$ is smaller than the length of the list.

For ease of understanding the above discussions, let us see an example shown in the following proposition (the proof of this proposition can be found in [15]). It basically says that given a probability space that is top-1 subgroup order preserving, a top-3 subgroup order sensitive surrogate loss function may not be consistent with the top-1 true loss.

**Proposition 10.** *Suppose there are three objects to be ranked. $\phi$ is a top-3 subgroup order sensitive loss function and the strict inequality $\phi(\mathbf{g}, (3,1,2)) < \phi(\mathbf{g}, (3,2,1))$ holds when $g_1 > g_2$. The probabilities of permutations are $p_{123} = p_1$, $p_{132} = 0$, $p_{213} = p_2$, $p_{231} = 0$, $p_{312} = 0$, $p_{321} = p_2$ respectively, where $p_1 > p_2$. Then $\phi$ is not consistent with the top-1 true loss.*

The above discussions imply that although the surrogate loss functions in existing listwise ranking methods are consistent with the permutation-level 0-1 loss (under a rigid condition), they may not be consistent with the top-$k$ true loss (under a mild condition). Therefore, it is necessary to modify these surrogate loss functions. We will make discussions on this in the next subsection.

### 4.3 Consistent surrogate loss functions

In [16], the surrogate loss functions in ListNet, RankCosine, and ListMLE have been proved to be permutation-level order sensitive. According to the discussion in the previous subsection, however, they may not be top-$k$ subgroup order sensitive, and therefore not consistent with the top-$k$ true loss. Even for the consistency with the permutation-level 0-1 loss, in order to guarantee these surrogate loss functions to be consistent, the requirement on the probability space may be too strong in some real scenarios. To tackle the challenge, it is desirable to modify these surrogate loss functions to make them top-$k$ subgroup order sensitive. Actually this is doable, and the modifications to the aforementioned surrogate loss functions are given as follows.

#### 4.3.1 Likelihood loss

The likelihood loss is the loss function used in ListMLE [16], which is defined as below,

$$\phi(\mathbf{g}(\mathbf{x}), y) = -\log P(y|\mathbf{x}; \mathbf{g}), \qquad \text{where } P(y|\mathbf{x}; \mathbf{g}) = \prod_{i=1}^{n} \frac{\exp(g(x_{y(i)}))}{\sum_{t=i}^{n} \exp(g(x_{y(t)}))}. \tag{9}$$

We propose replacing the permutation probability with the top-$k$ subgroup probability (which is also defined with the Luce model [11]) in the above definition:

$$P(y|\mathbf{x};\mathbf{g}) = \prod_{i=1}^{k} \frac{\exp(g(x_{y(i)}))}{\sum_{t=i}^{n} \exp(g(x_{y(t)}))}. \tag{10}$$

It can be proved that the modified loss is top-$k$ subgroup order sensitive (see [15]).

### 4.3.2 Cosine loss

The cosine loss is the loss function used in RankCosine [13], which is defined as follows,

$$\phi(\mathbf{g}(\mathbf{x}),y) = \frac{1}{2}(1 - \frac{\psi_y(\mathbf{x})^T \mathbf{g}(\mathbf{x})}{\|\psi_y(\mathbf{x})\|\|\mathbf{g}(\mathbf{x})\|}), \tag{11}$$

where the score vector of the ground truth is produced by a mapping function $\psi_y(\cdot) : R^d \to R$, which retains the order in a permutation, i.e., $\psi_y(x_{y(1)}) > \cdots > \psi_y(x_{y(n)})$.

We propose changing the mapping function as follows. Let the mapping function retain the order for the top $k$ positions of the ground truth permutation and assigns to all the remaining positions a small value (which is smaller than the score of any object ranked at the top-$k$ positions), i.e., $\psi_y(x_{y(1)}) > \cdots > \psi_y(x_{y(k)}) > \psi_y(x_{y(k+1)}) = \cdots = \psi_y(x_{y(n)}) = \epsilon$. It can be proved that after the modification, the cosine loss becomes top-$k$ subgroup order sensitive (see [15]).

### 4.3.3 Cross entropy loss

The cross entropy loss is the loss function used in ListNet [3], defined as follows,

$$\phi(\mathbf{g}(\mathbf{x}),y) = D(P(\pi|\mathbf{x};\psi_y)||P(\pi|\mathbf{x};\mathbf{g})), \tag{12}$$

where $\psi$ is a mapping function whose definition is similar to that in RankCosine, and $P(\pi|\mathbf{x};\psi_y)$ and $P(\pi|\mathbf{x};\mathbf{g})$ are the permutation probabilities in the Luce model.

We propose using a mapping function to modify the cross entropy loss in a similar way as in the case of the cosine loss[4] It can be proved that such a modification can make the surrogate loss function top-$k$ subgroup order sensitive (see [15]).

## 5 Experimental results

In order to validate the theoretical analysis in this work, we conducted some empirical study. Specifically, we used OHSUMED, TD2003, and TD2004 in the LETOR benchmark dataset [10] to perform some experiments. As evaluation measure, we adopted Normalized Discounted Cumulative Gain (N) at positions 1, 3, and 10, and Precision (P) at positions 1, 3, and 10.[5] It is obvious that these measures are top-$k$ related and are suitable to evaluate the ranking performance in top-$k$ ranking problems.

We chose ListMLE as example method since the likelihood loss has nice properties such as convexity, soundness, and linear computational complexity [16]. We refer to the new method that we obtained by applying the modifications mentioned in Section 4.3 as top-$k$ ListMLE. We tried different values of $k$ (i.e., $k$=1, 3, 10, and the exact length of the ranked list). Obviously the last case corresponds to the original likelihood loss in ListMLE.

Since the training data in LETOR is given in the form of multi-level ratings, we adopted the methods proposed in [16] to produce the ground truth ranked list. We then used stochastic gradient descent as the algorithm for optimization of the likelihood loss. As for the ranking model, we chose linear Neural Network, since the model has been widely used [3, 13, 16].

The experimental results are summarized in Tables 1-3.

| Methods | N@1 | N@3 | N@10 | P@1 | P@3 | P@10 |
|---|---|---|---|---|---|---|
| ListMLE | 0.548 | 0.473 | 0.446 | 0.642 | 0.582 | 0.495 |
| Top-1 ListMLE | 0.529 | 0.482 | 0.447 | 0.652 | 0.595 | 0.499 |
| Top-3 ListMLE | 0.535 | 0.484 | 0.445 | 0.671 | 0.608 | 0.504 |
| Top-10 ListMLE | 0.558 | 0.473 | 0.444 | 0.672 | 0.601 | 0.509 |

Table 1: Ranking accuracies on OHSUMED

| Methods | N/P@1 | N@3 | N@10 | P@3 | P@10 |
|---|---|---|---|---|---|
| ListMLE | 0.24 | 0.253 | 0.261 | 0.22 | 0.146 |
| Top-1 ListMLE | 0.4 | 0.329 | 0.314 | 0.3 | 0.176 |
| Top-3 ListMLE | 0.44 | 0.382 | 0.343 | 0.34 | 0.204 |
| Top-10 ListMLE | 0.5 | 0.410 | 0.378 | 0.38 | 0.22 |

Table 2: Ranking accuracies on TD2003

| Methods | N/P@1 | N@3 | N@10 | P@3 | P@10 |
|---|---|---|---|---|---|
| ListMLE | 0.4 | 0.351 | 0.356 | 0.284 | 0.188 |
| Top-1 ListMLE | 0.52 | 0.469 | 0.451 | 0.413 | 0.248 |
| Top-3 ListMLE | 0.506 | 0.456 | 0.458 | 0.417 | 0.261 |
| Top-10 ListMLE | 0.52 | 0.469 | 0.472 | 0.413 | 0.269 |

Table 3: Ranking accuracies on TD2004

| Methods | N@1 | N@3 | N@10 | P@1 | P@3 | P@10 |
|---|---|---|---|---|---|---|
| RankBoost | 0.497 | 0.472 | 0.435 | 0.604 | 0.586 | 0.495 |
| Ranking SVM | 0.495 | 0.464 | 0.441 | 0.633 | 0.592 | 0.507 |
| ListNet | 0.523 | **0.477** | **0.448** | 0.642 | **0.602** | **0.509** |
| RankCosine | 0.523 | 0.475 | 0.437 | 0.642 | 0.589 | 0.493 |
| Top-10 ListMLE | **0.558** | 0.473 | 0.444 | **0.672** | 0.601 | **0.509** |

Table 4: Ranking accuracies on OHSUMED

From the tables, we can see that with the modifications the ranking accuracies of ListMLE can be significantly boosted, in terms of all measures, on both TD2003 and TD2004. This clearly validates our theoretical analysis. On OHSUMED, all the loss functions achieve comparable performances. The possible explanation is that the probability space in OHSUMED is well formed such that it is order preserving for many different $k$ values.

Next, we take Top-10 ListMLE as an example to make comparison with some other baseline methods such as Ranking SVM [8], RankBoost [7], ListNet [3], and RankCosine [13]. The results are listed in Tables 4-6. We can see from the tables, Top-10 ListMLE achieves the best performance among all the methods on the TD2003 and TD2004 datasets in terms of almost all measures. On the OHSUMED dataset, it also performs fairly well as compared to the other methods. Especially for N@1 and P@1, it significantly outperforms all the other methods on all the datasets.

| Methods | N/P@1 | N@3 | N@10 | P@3 | P@10 |
|---|---|---|---|---|---|
| RankBoost | 0.26 | 0.270 | 0.285 | 0.24 | 0.178 |
| Ranking SVM | 0.42 | 0.378 | 0.341 | 0.34 | 0.206 |
| ListNet | 0.46 | 0.408 | 0.374 | 0.36 | **0.222** |
| RankCosine | 0.36 | 0.346 | 0.322 | 0.3 | 0.182 |
| Top-10 ListMLE | **0.5** | **0.410** | **0.378** | **0.38** | 0.22 |

Table 5: Ranking accuracies on TD2003

| Methods | N/P@1 | N@3 | N@10 | P@3 | P@10 |
|---|---|---|---|---|---|
| RankBoost | 0.48 | 0.463 | 0.471 | 0.404 | 0.253 |
| Ranking SVM | 0.44 | 0.409 | 0.420 | 0.351 | 0.225 |
| ListNet | 0.439 | 0.437 | 0.457 | 0.399 | 0.257 |
| RankCosine | 0.439 | 0.397 | 0.405 | 0.328 | 0.209 |
| Top-10 ListMLE | **0.52** | **0.469** | **0.472** | **0.413** | **0.269** |

Table 6: Ranking accuracies on TD2004

From the above experimental results, we can come to the conclusion that for real ranking applications like IR (where top-$k$ evaluation measures are widely used), it is better to use the top-$k$ true loss than the permutation-level 0-1 loss, and is better to use the modified surrogate loss functions than the original surrogate loss functions.

# 6  Conclusion

In this paper we have proposed a top-$k$ ranking framework, which can better describe real ranking applications like information retrieval. In the framework, the true loss is defined on the top-$k$ subgroup of permutations. We have derived the sufficient conditions for a surrogate loss function to be statistically consistent with the top-$k$ true loss. We have also discussed how to modify the loss functions in existing listwise ranking methods to make them consistent with the top-$k$ true loss. Our experiments have shown that with the proposed modifications, algorithms like ListMLE can significantly outperform their original version, and also many other ranking methods.

As future work, we plan to investigate the following issues. (1) we will empirically study the modified ListNet and RankCosine, to see whether their performances can also be significantly boosted in the top-$k$ setting. (2) We will also study the consistency of the pointwise and pairwise loss functions with the top-$k$ true loss.

## Footnotes

[1] iProspect Search Engine User Behavior Study, April 2006, http://www.iprospect.com/

[2] Note that the probability of a top-$k$ subgroup is defined as the sum of the probabilities of the permutations in the subgroup (cf., Definitions 6 and 7 in [3]).

[3]By trivial modifications, one can handle the case that $g_1$ or $g_2$ is infinite (cf. [17]).

[4]Note that in [3], a top-$k$ cross entropy loss was also proposed, by using the top-$k$ Luce model. However, it can be verified that the so-defined top-$k$ cross entropy loss is still permutation-level order sensitive, but not top-$k$ subgroup order sensitive. In other words, the proposed modification here is still needed.

[5]On datasets with only two ratings such as TD2003 and TD2004, N@1 equals P@1.

# References

[1] P. L. Bartlett, M. I. Jordan, and J. D. McAuliffe. Convexity, classification, and risk bounds. *Journal of the American Statistical Association*, 101:138–156, 2006.

[2] C. Burges, T. Shaked, E. Renshaw, A. Lazier, M. Deeds, N. Hamilton, and G. Hullender. Learning to rank using gradient descent. In *Proc. of ICML'05*, pages 89–96, 2005.

[3] Z. Cao, T. Qin, T. Y. Liu, M. F. Tsai, and H. Li. Learning to rank: From pairwise approach to listwise approach. In *Proc. of ICML'07*, pages 129–136, 2007.

[4] S. Clemencon and N. Vayatis. Ranking the best instances. *Journal of Machine Learning Research*, 8:2671–2699, 2007.

[5] D. Cossock and T. Zhang. Subset ranking using regression. In *Proc. of COLT*, pages 605–619, 2006.

[6] D. Cossock and T. Zhang. Statistical analysis of bayes optimal subset ranking. *Information Theory*, 54:5140–5154, 2008.

[7] Y. Freund, R. Iyer, R. E. Schapire, and Y. Singer. An efficient boosting algorithm for combining preferences. In *Proc. of ICML'98*, pages 170–178, 1998.

[8] R. Herbrich, T. Graepel, and K. Obermayer. Support vector vector learning for ordinal regression. In *Proc. of ICANN'99*, pages 97–102, 1999.

[9] P. Li, C. Burges, and Q. Wu. Mcrank: Learning to rank using multiple classification and gradient boosting. In *Advances in Neural Information Processing Systems 20(NIPS 07)*, pages 897–904, Cambridge, MA, 2008. MIT Press.

[10] T. Y. Liu, T. Qin, J. Xu, W. Y. Xiong, and H. Li. Letor: Benchmark dataset for research on learning to rank for information retrieval. In *LR4IR 2007, in conjunction with SIGIR 2007*, 2007.

[11] J. I. Marden, editor. *Analyzing and Modeling Rank Data*. Chapman and Hall, London, 1995.

[12] R. Nallapati. Discriminative models for information retrieval. In *Proc. of SIGIR'04*, pages 64–71, 2004.

[13] T. Qin, X.-D. Zhang, M.-F. Tsai, D.-S. Wang, T.-Y. Liu, and H. Li. Query-level loss functions for information retrieval. *Information processing and management*, 44:838–855, 2008.

[14] C. Rudin. Ranking with a p-norm push. In *Proc. of COLT*, pages 589–604, 2006.

[15] F. Xia, T. Y. Liu, and H. Li. Top-$k$ consistency of learning to rank methods. Technical report, Microsoft Research, MSR-TR-2009-139, 2009.

[16] F. Xia, T. Y. Liu, J. Wang, W. S. Zhang, and H. Li. Listwise approach to learning to rank - theory and algorithm. In *Proc. of ICML'08*, pages 1192–1199, 2008.

[17] T. Zhang. Statistical analysis of some multi-category large margin classification methods. *Journal of Machine Learning Research*, 5:1225–1251, 2004.

